# Online Discovery and Learning of Predictive State Representations

**Peter McCracken**
Department of Computing Science
University of Alberta
Edmonton, Alberta
Canada, T6G 2E8
peterm@cs.ualberta.ca

**Michael Bowling**
Department of Computing Science
University of Alberta
Edmonton, Alberta
Canada, T6G 2E8
bowling@cs.ualberta.ca

## Abstract

Predictive state representations (PSRs) are a method of modeling dynamical systems using only observable data, such as actions and observations, to describe their model. PSRs use predictions about the outcome of future tests to summarize the system state. The best existing techniques for discovery and learning of PSRs use a Monte Carlo approach to explicitly estimate these outcome probabilities. In this paper, we present a new algorithm for discovery and learning of PSRs that uses a gradient descent approach to compute the predictions for the current state. The algorithm takes advantage of the large amount of structure inherent in a valid prediction matrix to constrain its predictions. Furthermore, the algorithm can be used online by an agent to constantly improve its prediction quality; something that current state of the art discovery and learning algorithms are unable to do. We give empirical results to show that our constrained gradient algorithm is able to discover core tests using very small amounts of data, and with larger amounts of data can compute accurate predictions of the system dynamics.

## 1 Introduction

Representations of state in dynamical systems fall into three main categories. Methods like $k$-order Markov models attempt to identify state by remembering what has happened in the past. Methods such as partially observable Markov decision processes (POMDPs) identify state as a distribution over postulated base states. A more recently developed group of algorithms, known as *predictive representations*, identify state in dynamical systems by predicting what will happen in the future. Algorithms following this paradigm include observable operator models [1], predictive state representations [2, 3], TD-Nets [4] and TPSRs [5]. In this research we focus on predictive state representations (PSRs). PSRs are completely grounded in data obtained from the system, and they have been shown to be at least as general and as compact as other methods, like POMDPs [3].

Until recently, algorithms for discovery and learning of PSRs could be used only in special cases. They have required explicit control of the system using a reset action [6, 5], or have required the incoming data stream to be generated using an open-loop policy [7].

The algorithm presented in this paper does not require a reset action, nor does it make any assumptions about the policy used to generate the data stream. Furthermore, we focus on the *online* learning problem, *i.e.*, how can an estimate of the *current* state vector and parameters be maintained and improved during a single pass over a string of data. Like the myopic gradient descent algorithm [8], the algorithm we propose uses a gradient approach to move its predictions closer to its empirical observations; however, our algorithm also takes advantage of known constraints on valid test predictions. We show that this constrained gradient approach is capable of discovering a set of core tests quickly, and also of making online predictions that improve as more data is available.

## 2 Predictive State Representations

Predictive state representations (PSRs) were introduced by Littman *et al.* [2] as a method of modeling discrete-time, controlled dynamical systems. They possess several advantages over other popular models such as POMDPs and $k$-order Markov models, foremost being their ability to be learned entirely from sensorimotor data, requiring only a prior knowledge of the set of actions, $\mathcal{A}$, and observations, $\mathcal{O}$.

**Notation.** An agent in a dynamical system experiences a sequence of action-observation pairs, or $ao$ pairs. The sequence of $ao$ pairs the agent has already experienced, beginning at the first time step, is known as a *history*. For instance, the history $h^n = a^1 o^1 a^2 o^2 \ldots a^n o^n$ of length $n$ means that the agent chose action $a^1$ and perceived observation $o^1$ at the first time step, after which the agent chose $a^2$ and perceived $o^2$, and so on[1]. A *test* is a sequence of $ao$ pairs that begins immediately after a history. A test is said to succeed if the observations in the sequence are observed in order, given that the actions in the sequence are chosen in order. For instance, the test $t = a_1 o_1 a_2 o_2$ succeeds if the agent observes $o_1$ followed by $o_2$, given that it performs actions $a_1$ followed by $a_2$. A test fails if the action sequence is taken but the observation sequence is not observed. A prediction about the outcome of a test $t$ depends on the history $h$ that preceded it, so we write predictions as $p(t|h)$, to represent the probability of $t$ succeeding after history $h$. For test $t$ of length $n$, we define a prediction $p(t|h)$ as $\prod_{i=1}^{n} \Pr(o_i|a_1 o_1 \ldots a_i)$. This definition is equivalent to the usual definition in the PSR literature, but makes it explicit that predictions are independent of the policy used to select actions. The special length zero test is called $\varepsilon$. If $T$ is a set of tests and $H$ is a set of histories, $p(t|h)$ is a single value, $p(T|h)$ is a row vector containing $p(t_i|h)$ for all tests $t_i \in T$, $p(t|H)$ is a column vector containing $p(t|h_j)$ for all histories $h_j \in H$, and $p(T|H)$ is a matrix containing $p(t_i|h_j)$ for all $t_i \in T$ and $h_j \in H$.

**PSRS.** The fundamental principle underlying PSRs is that in most systems there exists a set of tests, $Q$, that at any history are a sufficient statistic for determining the probability of success for all possible tests. This means that for any test $t$ there exists a function $f_t$ such that $p(t|h) = f_t(p(Q|h))$. In this paper, we restrict our discussion of PSRs to *linear* PSRs, in which the function $f_t$ is a linear function of the tests in $Q$. Thus, $p(t|h) = p(Q|h)m_t$, where $m_t$ is a column vector of weights. The tests in $Q$ are known as *core tests*, and determining which tests are core tests is known as the *discovery* problem. In addition to $Q$, it will be convenient to discuss the set of one-step extensions of $Q$. A one-step extension of a test $t$ is a test $aot$, that prefixes the original test with a single $ao$ pair. The set of all one-step extensions of $Q \cup \{\varepsilon\}$ will be called $X$.

The state vector of a PSR at time $i$ is the set of predictions $p(Q|h^i)$. At each time step, the

state vector is updated by computing, for each $q_j \in Q$:

$$p(q_j|h^i) = \frac{p(a^i o^i q_j|h^{i-1})}{p(a^i o^i|h^{i-1})} = \frac{p(Q|h^{i-1})m_{a^i o^i q_j}}{p(Q|h^{i-1})m_{a^i o^i}}$$

Thus, in order to update the PSR at each time step, the vector $m_t$ must be known for each test $t \in X$. This set of update vectors, that we will call $m_X$, are the parameters of the PSR, and estimation of these parameters is known as the *learning* problem.

## 3 Constrained Gradient Learning of PSRs

The goal of this paper is to develop an online algorithm for discovering and learning a PSR without the necessity of a reset action. To be online, the algorithm must always have an estimate of the current state vector, $p(Q|h^i)$, and estimates of the parameters $m_X$. In this section, we introduce our constrained gradient approach to solving this problem. A more complete explanation of this algorithm can be found in an expanded version of this work [9]. To begin, in Section 3.1, we will assume that the set of core tests $Q$ is given to the algorithm; we describe how $Q$ can be estimated online in Section 3.2.

### 3.1 Learning the PSR Parameters

The approach to learning taken by the constrained gradient algorithm is to approximate the matrix $p(T|H)$, for a selected set of tests $T$ and histories $H$. We first discuss the proper selection of $T$ and $H$, and then describe how this matrix can be constructed online. Finally, we show how the current PSR is extracted from the matrix.

**Tests and Histories.** At a minimum, $T$ must contain the union of $Q$ and $X$, since $Q$ is required to create the state vector and $X$ is required to compute $m_X$. However, as will be explained in the next section, these tests are not sufficient to take full advantage of the structure in a prediction matrix. The constrained gradient algorithm requires the tests in $T$ to satisfy two properties:

1. If $tao \in T$ then $t \in T$
2. If $tao_i \in T$ then $tao_j \in T \quad \forall o_j \in \mathcal{O}$

To build a valid set of tests, $T$ is initialized to $Q \cup X$. Tests are iteratively added to $T$ until it satisfies both of the above properties.

All histories in $H$ are histories that have been experienced by the agent. The current history, $h^i$, must always be in $H$ in order to make online predictions, and also to compute $h^{i+1}$. The only other requirement of $H$ is that it contain sufficient histories to compute the linear functions $m_t$ for the tests in $T$ (see Section 3.1). Our strategy is impose a bound $N$ on the size of $H$, and to restrict $H$ to the $N$ most recent histories encountered by the agent. When a new data point is seen and a new row is added to the matrix, the oldest row in the matrix is "forgotten." In addition to restricting the size of $H$, forgetting old rows has the side-effect that the rows estimated using the least amount of data are removed from the matrix, and no longer affect the computation of $m_X$.

**Constructing the Prediction Matrix.** The approach used to build the matrix $p(T|H)$ is to estimate and append a new row, $p(T|h^i)$, after each new $a^i o^i$ pair is encountered. Once a row has been added, it is never changed. To initialize the algorithm, the first row of the matrix $p(T|h^0)$, is set to uniform probabilities.[2] The creation of the new row is performed in two stages: a row estimation stage, and a gradient descent stage.

Both stages take advantage of four constraints on the predictions $p(T|h)$ in order to be a valid row in the prediction matrix:

1. Range: $0 \leq p(t|h) \leq 1$
2. Null Test: $p(\varepsilon|h) = 1$
3. Internal Consistency: $p(t|h) = \sum_{o_j \in \mathcal{O}} p(tao_j|h) \quad \forall a \in \mathcal{A}$
4. Conditional Probability: $p(t|hao) = p(aot|h)/p(ao|h) \quad \forall a \in \mathcal{A}, o \in \mathcal{O}$

The range constraint restricts the entries in the matrix to be valid probabilities. The null test constraint defines the value of the null test. The internal consistency constraint ensures that the probabilities within a single row form valid probability distributions. The conditional probability constraint is required to maintain consistency between consecutive rows of the matrix.

Consider time $i-1$ so that the last row of $p(T|H)$ is $h^{i-1}$. After action $a^i$ is taken and observation $o^i$ is seen, a new row for history $h^i = h^{i-1}a^io^i$ must be added to the matrix. First, as much of the new row as possible is computed using the conditional probability constraint, and the predictions for history $h^{i-1}$. For all tests $t \in T$ for which $a^io^it \in T$:

$$p(t|h^i) \leftarrow \frac{p(a^io^it|h^{i-1})}{p(a^io^i|h^{i-1})}$$

Because $X \subset T$, it is guaranteed that $p(Q|h^i)$ is estimated in this step.

The second phase of adding a new row is to compute predictions for the tests $t \in T$ for which $a^io^it \notin T$. An estimate of $p(t|h^i)$ can be found by computing $p(Q|h^i)m_t$ for an appropriate $m_t$, using the PSR assumption that any prediction is a linear combination of core test predictions. Regression is used to find a vector $m_t$ that minimizes $||p(Q|H)m_t - p(t|H)||^2$. At this stage, the entire row for $h^i$ has been estimated. The regression step can create probabilities that violate the range and normalization properties of a valid prediction. To enforce the range property, any predictions that are less than 0 are set to a small positive value[3]. Then, to ensure internal consistency within the row, the normalization property is enforced by setting predictions:

$$p(tao_j|h^i) \leftarrow \frac{p(t|h^i)p(tao_j|h^i)}{\sum_{o_i \in \mathcal{O}} p(tao_i|h^i)} \quad \forall o_j \in \mathcal{O}$$

This preserves the ratio among sibling predictions and creates a valid probability distribution from them. The normalization is performed by normalizing shorter tests first, which guarantees that a set of tests are not normalized to a value that will later change. The length one tests are normalized to sum to 1.

The gradient descent stage of estimating a new row moves the constraint-generated predictions in the direction of the gradient created by the new observation. Any prediction $p(tao|h^i)$ whose test $tao$ is successfully executed over the next several time steps is updated using $p(tao|h^i) \leftarrow (1-\alpha)p(tao|h^i) + \alpha(p(t|h^i))$, for some learning rate $0 \leq \alpha \leq 1$. Note that this learning rule is a temporal difference update; prediction values are adjusted toward the value of their parent.[4] The update is accomplished by adding an appropriate positive value to $p(tao|h^i)$ and then running the normalization procedure on the row. The value is computed such that after normalization, $p(tao|h^i)$ contains the desired value. Tests

that are unsuccessfully executed (*i.e.* their action sequence is executed but their observation sequence is not observed) will have their probability reduced due to this re-normalization step. The learning parameter, $\alpha$, is decayed throughout the learning process.

**Extracting the PSR.** Once a new row for $h^i$ is estimated, the current PSR state vector is $p(Q|h^i)$. The parameters $m_X$ can be found by using the output of the regression from the second phase, above. Thus, at every time step, the current best estimated PSR of the system is available.

### 3.2 Discovery of Core Tests

In the previous section, we assumed that the set of core tests was given to the algorithm. In general, though, $Q$ is not known. A rudimentary, but effective, method of finding core tests is to choose tests whose corresponding columns of the matrix $p(T|H)$ are most linearly unrelated to the set of core tests already selected. Call the set of selected core tests $\widehat{Q}$. The condition number of the matrix $p(\{\widehat{Q}, t\}|H)$ is an indication of the linear relatedness of test $t$; if it is well-conditioned, the test is likely to be linearly independent. To choose core tests, we find the test $t$ in $X$ whose matrix $p(\{\widehat{Q}, t\}|H)$ is most well-conditioned. If the condition number of that test is below a threshold parameter, it is chosen as a new core test. The process can be repeated until no test can be added to $\widehat{Q}$ without surpassing the threshold. Because candidate tests are selected from $X$, the discovered set $\widehat{Q}$ will be a regular form PSR [10].

The set $\widehat{Q}$ is initialized to $\{\varepsilon\}$. The above core test selection procedure runs after every $N$ data points are seen, where $N$ is the maximum number of histories kept in $H$. After each new core test is selected, $T$ is augmented with the one-step extensions of the new test, as well as any other tests needed to satisfy the rules in Section 3.1.

## 4 Experiments and Results

The goal of the constrained gradient algorithm is to choose a correct set of core tests and to make accurate, online predictions. In this section, we show empirical results that the algorithm is capable of these goals. We also show offline results, in order to compare our results with the suffix-history algorithm [7]. A more thorough suite of experiments can be found in an expanded version of this work [9].

We tested our algorithm on the same set of problems from Cassandra's POMDP page [11] used as the test domain in other PSR trials [8, 6, 7]. For each problem, 10 trials were run, with different training sequences and test sequences used for each trial. The sequences were generated using a uniform random policy over actions. The error for each history $h^i$ was computed using the error measure $\frac{1}{|O|} \sum_{o_j \in O} (p(a^{i+1} o_j | h^i) - \hat{p}(a^{i+1} o_j | h^i))^2$ [7]. This measures the mean error in the one-step tests involving the action that was actually taken at step $i + 1$.

The same parameterization of the algorithm was used for all domains. The size bound on $H$ was set to 1000, and the condition threshold for adding new tests was 10. The learning parameter $\alpha$ was initialized to 1 and halved every 100,000 time steps. The core test discovery procedure was run every 1000 data points.

### 4.1 Discovery Results

In this section, we examine the success of the constrained gradient algorithm at discovering core tests. Table 1 shows, for each test domain, the true number of core tests for the

Table 1: The number of core tests found by the constrained gradient algorithm. Data for the suffix-history algorithm [7] is repeated here for comparison. See text for explanation.

| Domain | | Constrained Gradient | | | Suffix-History | |
| Name | $|Q|$ | $|\widehat{Q}|$ | Correct | # Data | $|\widehat{Q}|$/Correct | # Data |
|---|---|---|---|---|---|---|
| Float Reset | 5 | 6.1 | 4.5 | 4000 | - | - |
| Tiger | 2 | 4.0 | 2.0 | 1000 | 2 | 4000 |
| Paint | 2 | 2.6 | 2.0 | 4000 | 2 | 4000 |
| Shuttle | 7 | 8.7 | 7.0 | 2000 | 7 | 1024000 |
| 4x3 Maze | 10 | 10.4 | 8.6 | 2000 | 9 | 1024000 |
| Cheese Maze | 11 | 12.1 | 9.6 | 1000 | 9 | 32000 |
| Bridge Repair | 5 | 7.2 | 5.0 | 1000 | 5 | 1024000 |
| Network | 7 | 4.7 | 4.5 | 2000 | 3 | 2048000 |

dynamical system ($|Q|$), the number of core tests selected by the constrained gradient algorithm ($|\widehat{Q}|$), and how many of the selected core tests were *actually* core tests (*Correct*). The results are averaged over 10 trials. Table 1 also shows the time step at which the last core test was chosen (*# Data*). In all domains, the algorithm found a majority of the core tests after only several thousand data points; in several cases, the core tests were found after only a single run of the core test selection procedure.

Table 1 also shows discovery results published for the suffix-history algorithm [7]. All of the core tests found by the suffix-history algorithm were true core tests. In all cases except the 4x3 Maze, the constrained gradient algorithm was able to find at least as many core tests as the suffix-history method, and required significantly less data. To be fair, the suffix-history algorithm uses a conservative approach of selecting core tests, and therefore requires more data. The constrained gradient algorithm chooses tests that give an early indication of being linearly independent. Therefore, the constrained gradient finds most, or all, core tests extremely quickly, but can also choose tests that are not linearly independent.

## 4.2  Online and Offline Results

Figure 1 shows the performance of the constrained gradient approach, in online and offline settings. The question answered by the online experiments is: *How accurately can the constrained gradient algorithm predict the outcome of the next time step?* At each time $i$, we measured the error in the algorithm's predictions of $p(a^{i+1}o_j|h^i)$ for each $o_j \in O$. The 'Online' plot in Figure 1 shows the mean online error from the previous 1000 time steps.

The question posed for the offline experiments was: *What is the long-term performance of the PSRs learned by the constrained gradient algorithm?* To test this, we stopped the learning process at different points in the training sequence and computed the current PSR. The initial state vector for the offline tests was set to the column means of $p(\widehat{Q}|H)$, which approximates the state vector of the system's stationary distribution. In Figure 1, the 'Offline' plot shows the mean error of this PSR on a test sequence of length 10,000. The offline and online performances of the algorithm are very similar. This indicates that, after a given amount of data, the immediate error on the next observation and the long-term error of the generated PSR are approximately the same. This result is encouraging because it implies that the PSR remains stable in its predictions, even in the long term.

Previously published [7] performance results for the suffix-history algorithm are also shown in Figure 1. A direct comparison between the performance of the two algorithms is somewhat inappropriate, because the suffix-history algorithm solves the 'batch' problem and is able to make multiple passes over the data stream. However, the comparison does show that

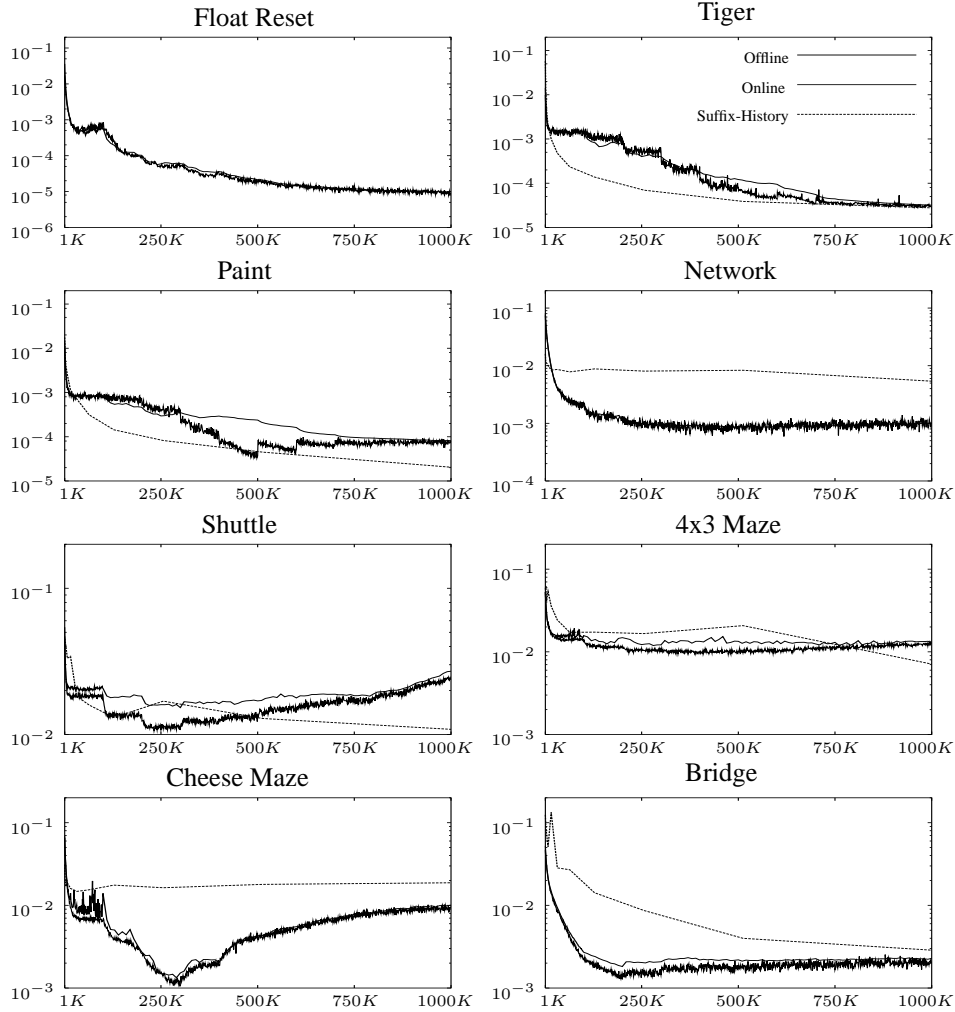

Figure 1: The PSR error on the test domains. The x-axis is the length of the sequence used for training, which ranges from 1,000 to 1,000,000. The y-axis shows the mean error on the one-step predictions (Online) or on a test sequence (Offline and Suffix-History). The results for Suffix-History are repeated from previous work [7]. See text for explanations.

the constrained gradient approach is competitive with current PSR learning algorithms.

The performance plateau in the 4x3 Maze and Network domains is unsurprising, because in these domains only a subset of the correct core tests were found (see Table 1). The plateau in the Bridge domain is more concerning, because in this domains all of the correct core tests were found. We suspect this may be due to a local minimum in the error space; more tests need to be performed to investigate this phenomenon.

## 5  Future Work and Conclusion

We have demonstrated that the constrained gradient algorithm can do online learning and discovery of predictive state representations from an arbitrary stream of experience. We

have also shown that it is competitive with the alternative batch methods. There are still a number of interesting directions for future improvement.

In the current method of core test selection, the condition of the core test matrix $p(\widehat{Q}|H)$ is important. If the matrix becomes ill-conditioned, it prevents new core tests from becoming selected. This can happen if the true core test matrix $p(Q|H)$ is poorly conditioned (because some core tests are similar), or if incorrect core tests are added to $\widehat{Q}$. To prevent this problem, there needs to be a mechanism for removing chosen core tests if they turn out to be linearly dependent. Also, the condition threshold should be gradually increased during learning, to allow more obscure core tests to be selected.

Another interesting modification to the algorithm is to replace the current multi-step estimation of new rows with a single optimization. We want to simultaneously minimize the regression error and next observation error subject to the constraints on valid predictions. This optimization could be solved with quadratic programming.

To date, the constrained gradient algorithm is the only PSR algorithm that takes advantage of the sequential nature of the data stream experienced by the agent, and the constraints such a sequence imposes on the system. It handles the lack of a reset action without partitioning histories. Also, at the end of learning the algorithm has an estimate of the current state, instead of a prediction of the initial distribution or a stationary distribution over states. Empirical results show that, while there is room for improvement, the constrained gradient algorithm is competitive in both discovery and learning of PSRs.

## Footnotes

[1] Much of the notation used in this paper is adopted from Wolfe *et al.* [7]. Here we use the notation that a superscript $a^i$ or $o^i$ indicates the time step of an action or observation, and a subscript $a_i$ or $o_i$ indicates that the action or observation is a particular element of the set $\mathcal{A}$ or $\mathcal{O}$.

[2] Each $p(t|h^0)$ is set to $1/|\mathcal{O}|^k$, where $k$ is the length of test $t$.

[3]Setting values to zero can cause division by zero errors, if the prediction probability was not actually supposed to be zero.

[4]When the algorithm is used online, looking forward into the stream is impossible. In this case, we maintain a buffer of $ao$ pairs between the current time step and the histories that are added to the prediction matrix. The length of the buffer is the length of the longest test in $T$. To compute the predictions for the current time step, we iteratively update the PSR using the buffered data.

## References

[1] Herbert Jaeger. Observable operator models for discrete stochastic time series. *Neural Computation*, 12(6):1371–1398, 2000.

[2] Michael Littman, Richard Sutton, and Satinder Singh. Predictive representations of state. In *Advances in Neural Information Processing Systems 14 (NIPS)*, pages 1555–1561, 2002.

[3] Satinder Singh, Michael R. James, and Matthew R. Rudary. Predictive state representations: A new theory for modeling dynamical systems. In *Uncertainty in Artificial Intelligence: Proceedings of the Twentieth Conference (UAI)*, pages 512–519, 2004.

[4] Richard Sutton and Brian Tanner. Temporal-difference networks. In *Advances in Neural Information Processing Systems 17*, pages 1377–1384, 2005.

[5] Matthew Rosencrantz, Geoff Gordon, and Sebastian Thrun. Learning low dimensional predictive representations. In *Twenty-First International Conference on Machine Learning (ICML)*, 2004.

[6] Michael R. James and Satinder Singh. Learning and discovery of predictive state representations in dynamical systems with reset. In *Twenty-First International Conference on Machine Learning (ICML)*, 2004.

[7] Britton Wolfe, Michael R. James, and Satinder Singh. Learning predictive state representations in dynamical systems without reset. In *Twenty-Second International Conference on Machine Learning (ICML)*, 2005.

[8] Satinder Singh, Michael Littman, Nicholas Jong, David Pardoe, and Peter Stone. Learning predictive state representations. In *Twentieth International Conference on Machine Learning (ICML)*, pages 712–719, 2003.

[9] Peter McCracken. An online algorithm for discovery and learning of prediction state representations. Master's thesis, University of Alberta, 2005.

[10] Eric Wiewiora. Learning predictive representations from a history. In *Twenty-Second International Conference on Machine Learning (ICML)*, 2005.

[11] Anthony Cassandra. Tony's POMDP file repository page. http://www.cs.brown.edu/research/ai/pomdp/examples/index.html, 1999.
